# Multi-Robot Negotiation: Approximating the Set of Subgame Perfect Equilibria in General-Sum Stochastic Games

**Chris Murray**
Carnegie Mellon University
5000 Forbes Avenue
Pittsburgh, PA 15213

**Geoffrey J. Gordon**
Carnegie Mellon University
5000 Forbes Avenue
Pittsburgh, PA 15213

## Abstract

In real-world planning problems, we must reason not only about our own goals, but about the goals of other agents with which we may interact. Often these agents' goals are neither completely aligned with our own nor directly opposed to them. Instead there are opportunities for cooperation: by joining forces, the agents can all achieve higher utility than they could separately. But, in order to cooperate, the agents must negotiate a mutually acceptable plan from among the many possible ones, and each agent must trust that the others will follow their parts of the deal. Research in multi-agent planning has often avoided the problem of making sure that all agents have an incentive to follow a proposed joint plan. On the other hand, while game theoretic algorithms handle incentives correctly, they often don't scale to large planning problems. In this paper we attempt to bridge the gap between these two lines of research: we present an efficient game-theoretic approximate planning algorithm, along with a negotiation protocol which encourages agents to compute and agree on joint plans that are fair and optimal in a sense defined below. We demonstrate our algorithm and protocol on two simple robotic planning problems.[1]

## 1  INTRODUCTION

We model the multi-agent planning problem as a general-sum stochastic game with cheap talk: the agents observe the state of the world, discuss their plans with each other, and then simultaneously select their actions. The state and actions determine a one-step reward for each player and a distribution over the world's next state, and the process repeats.

While talking allows the agents to coordinate their actions, it cannot by itself solve the problem of trust: the agents might lie or make false promises. So, we are interested in planning algorithms that find *subgame-perfect Nash equilibria*. In a subgame-perfect equilibrium, every deviation from the plan is deterred by the threat of a suitable punishment, and every threatened punishment is believable. To find these equilibria, planners must reason about their own and other agents' incentives to deviate: if other agents have incentives to deviate then I can't trust them, while if I have an incentive to deviate, they can't trust me.

In a given game there may be many subgame-perfect equilibria with widely differing payoffs: some will be better for some agents, and others will be better for other agents. It is generally not feasible to compute all equilibria [1], and even if it were, there would be no obvious way

to select one to implement. It does not make sense for the agents to select an equilibrium without consulting one another: there is no reason that agent A's part of one joint plan would be compatible with agent B's part of another joint plan. Instead the agents must negotiate, computing and proposing equilibria until they find one which is acceptable to all parties.

This paper describes a planning algorithm and a negotiation protocol which work together to ensure that the agents compute and select a subgame-perfect Nash equilibrium which is both approximately *Pareto-optimal* (that is, its value to any single agent cannot be improved very much without lowering the value to another another agent) and approximately *fair* (that is, near the so-called *Nash bargaining point*). Neither the algorithm nor the protocol is guaranteed to work in all games; however, they are guaranteed correct when they are applicable, and applicability is easy to check. In addition, our experiments show that they work well in some realistic situations. Together, these properties of fairness, enforceability, and Pareto optimality form a strong solution concept for a stochastic game. The use of this definition is one characteristic that distinguishes our work from previous research: ours is the first efficient algorithm that we know of to use such a strong solution concept for stochastic games.

Our planning algorithm performs dynamic programming on a set-based value function: for $P$ players, at a state $s$, $V \in \mathbf{V}(s) \subset \mathbb{R}^P$ is an estimate of the value the players can achieve. We represent $\mathbf{V}(s)$ by sampling points on its convex hull. This representation is *conservative*, i.e., guarantees that we find a subset of the true $\mathbf{V}^*(s)$. Based on the sampled points we can efficiently compute one-step backups by checking which joint actions are enforceable in an equilibrium.

Our negotiation protocol is based on a multi-player version of Rubinstein's bargaining game. Players together enumerate a set of equilibria, and then take turns proposing an equilibrium from the set. Until the players agree, the protocol ends with a small probability $\epsilon$ after each step and defaults to a low-payoff equilibrium; the fear of this outcome forces players to make reasonable offers.

## 2 BACKGROUND

### 2.1 STOCHASTIC GAMES

A stochastic game represents a multi-agent planning problem in the same way that a Markov Decision Process [2] represents a single-agent planning problem. As in an MDP, transitions in a stochastic game depend on the current state and action. Unlike MDPs, the current (joint) action is a vector of individual actions, one for each player. More formally, a general-sum stochastic game $G$ is a tuple $(S, s_{\text{start}}, P, A, T, R, \gamma)$. $S$ is a set of states, and $s_{\text{start}} \in S$ is the start state. $P$ is the number of players. $A = A_1 \times A_2 \times \ldots \times A_P$ is the finite set of joint actions. We deal with fully observable stochastic games with perfect monitoring, where all players can observe previous joint actions. $T : S \times A \mapsto P(S)$ is the transition function, where $P(S)$ is the set of probability distributions over $S$. $R : S \times A \mapsto \mathbb{R}^P$ is the reward function. We will write $R_p(s, a)$ for the $p$th component of $\mathbf{R}(s, a)$. $\gamma \in [0, 1)$ is the discount factor. Player $p$ wants to maximize her *discounted total value* for the observed sequence of states and joint actions $s_1, a_1, s_2, a_2, \ldots$, $V_p = \sum_{t=1}^{\infty} \gamma^{t-1} R_p(s_t, a_t)$. A stationary policy for player $p$ is a function $\pi_p : S \mapsto P(A_p)$. A stationary joint policy is a vector of policies $\pi = (\pi_1, \ldots, \pi_P)$, one for each player. A nonstationary policy for player $p$ is a function $\pi_p : (\cup_{t=0}^{\infty} (S \times A)^t \times S) \mapsto P(A_p)$ which takes a history of states and joint actions and produces a distribution over player $p$'s actions; we can define a nonstationary joint policy analogously. For any nonstationary joint policy, there is a stationary policy that achieves the same value at every state [3].

The value function $V_p^{\pi} : S \mapsto \mathbb{R}$ gives expected values for player $p$ under joint policy $\pi$. The *value vector* at state $s$, $\mathbf{V}^{\pi}(s)$, is the vector with components $V_p^{\pi}(s)$. (For a nonstationary policy $\pi$ we will define $V_p^{\pi}(s)$ to be the value if $s$ were the start state, and $V_p^{\pi}(h)$ to be the value after observing history $h$.) A vector $\mathbf{V}$ is *feasible* at state $s$ if there is a $\pi$ for which $\mathbf{V}^{\pi}(s) = \mathbf{V}$, and we will say that $\pi$ *achieves* $\mathbf{V}$.

We will assume *public randomization*: the agents can sample from a desired joint action distribution in such a way that everyone can verify the outcome. If public randomization is not directly available, there are cryptographic protocols which can simulate it [4]. This assumption means that the set of feasible value vectors is convex, since we can roll a die at the first time step to choose from a set of feasible policies.

## 2.2 EQUILIBRIA

While optimal policies for MDPs can be determined exactly via various algorithms such as linear programming [2], it isn't clear what it means to find an optimal policy for a general sum stochastic game. So, rather than trying to determine a unique optimal policy, we will define a set of reasonable policies: the Pareto-dominant subgame-perfect Nash equilibria.

A (possibly nonstationary) joint policy $\pi$ is a *Nash equilibrium* if, for each individual player, no unilateral deviation from the policy would increase that player's expected value for playing the game. Nash equilibria can contain *incredible threats*, that is, threats which the agents have no intention of following through on. To remove this possibility, we can define the *subgame-perfect Nash equilibria*. A policy $\pi$ is a subgame-perfect Nash equilibrium if it is a Nash equilibrium in every possible subgame: that is, if there is no incentive for any player to deviate after observing any history of joint actions.

Finally, consider two policies $\pi$ and $\phi$. If $V_p^\pi(s_{\text{start}}) \geq V_p^\phi(s_{\text{start}})$ for all players $p$, and if $V_p^\pi(s_{\text{start}}) > V_p^\phi(s_{\text{start}})$ for at least one $p$, then we will say that $\pi$ *Pareto dominates* $\phi$. A policy which is not Pareto dominated by any other policy is *Pareto optimal*.

## 2.3 RELATED WORK

Littman and Stone [5] give an algorithm for finding Nash equilibria in two-player repeated games. Hansen et al. [6] show how to eliminate very-weakly-dominated strategies in partially observable stochastic games. Doraszelski and Judd [7] show how to compute Markov perfect equilibria in continuous-time stochastic games. The above papers use solution concepts much weaker than Pareto-dominant subgame-perfect equilibrium, and do not address negotiation and coordination. Perhaps the closest work to the current paper is by Brafman and Tennenholtz [8]: they present learning algorithms which, in repeated self-play, find Pareto-dominant (but not subgame-perfect) Nash equilibria in matrix and stochastic games. By contrast, we consider a single play of our game, but allow "cheap talk" beforehand. And, our protocol encourages arbitrary algorithms to agree on Pareto-dominant equilibria, while their result depends strongly on the self-play assumption.

### 2.3.1 FOLK THEOREMS

In any game, each player can guarantee herself an expected discounted value regardless of what actions the other players takes. We call this value the *safety value*. Suppose that there is a stationary subgame-perfect equilibrium which achieves the safety value for both players; call this the safety equilibrium policy.

Suppose that, in a repeated game, some stationary policy $\pi$ is better for both players than the safety equilibrium policy. Then we can build a subgame-perfect equilibrium with the same payoff as $\pi$: start playing $\pi$, and if someone deviates, switch to the safety equilibrium policy. So long as $\gamma$ is sufficiently large, no rational player will want to deviate. This is the *folk theorem for repeated games*: any feasible value vector which is strictly better than the safety values corresponds to a subgame-perfect Nash equilibrium [9]. (The proof is slightly more complicated if there is no safety equilibrium policy, but the theorem holds for any repeated game.)

There is also a folk theorem for general stochastic games [3]. This theorem, while useful, is not strong enough for our purposes: it only covers discount factors $\gamma$ which are so close to 1 that the players don't care which state they wind up in after a possible deviation. In most practical stochastic games, discount factors this high are unreasonably patient. When $\gamma$ is significantly less than 1, the set of equilibrium vectors can change in strange ways as we change $\gamma$ [10].

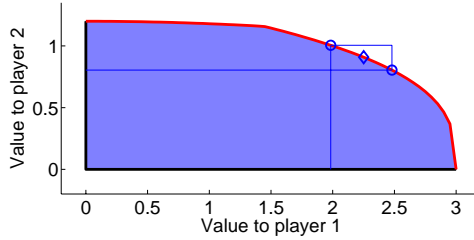

Figure 1: Equilibria of a Rubinstein game with $\gamma = 0.8$. Shaded area shows feasible value vectors $(U_1(x), U_2(x))$ for outcomes $x$. Right-hand circle corresponds to equilibrium when player 1 moves first, left-hand circle when player 2 moves first. The Nash point is at $\diamond$.

### 2.3.2   RUBINSTEIN'S GAME

Rubinstein [11] considered a game where two players divide a slice of pie. The first player offers a division $x, 1 - x$ to the second; the second player either accepts the division, or refuses and offers her own division $1 - y, y$. The game repeats until some player accepts an offer or until either player gives up. In the latter case neither player gets any pie. Rubinstein showed that if player $p$'s utility for receiving a fraction $x$ at time $t$ is $U_p(x, t) = \gamma^t U_p(x)$ for a discount factor $0 \leq \gamma < 1$ and an appropriate time-independent utility function $U_p(x) \geq 0$, then rational players will agree on a division near the so-called *Nash bargaining point*. This is the point which maximizes the product of the utilities that the players gain by cooperating, $U_1(x)U_2(1 - x)$. As $\gamma \uparrow 1$, the equilibrium will approach the Nash point. See Fig. 1 for an illustration. For three or more players, a similar result holds where agents take turns proposing multi-way divisions of the pie [12]. See the technical report [13] for more detail on the multi-player version of Rubinstein's game and the Nash bargaining point.

## 3   NEGOTIATION PROTOCOL

The Rubinstein game implicitly assumes that the result of a failure to cooperate is known to all players: nobody gets any pie. The multi-player version of the game assumes in addition that giving one player a share of the pie doesn't force us to give a share to any other player. Neither of these properties holds for general stochastic games. They are, however, easy to check, and often hold or can be made to hold for planning domains of interest.

So, we will assume that the players have agreed beforehand on a subgame-perfect equilibrium $\pi^{\text{dis}}$, called the *disagreement policy*, that they will follow in the event of a negotiation failure. In addition, for games with three or more players, we will assume that each player can unilaterally reduce her own utility by any desired amount without affecting other players' utilities.

Given these assumptions, our protocol proceeds in two phases (pseudocode is given in the technical report [13]. In the first phase agents compute subgame-perfect equilibria and take turns revealing them. On an agent's turn she either reveals an equilibrium or passes; if all agents pass consecutively, the protocol proceeds to the second phase. When an agent states a policy $\pi$, the other agents verify that $\pi$ is a subgame-perfect equilibrium and calculate its payoff vector $\mathbf{V}^\pi(s_{\text{start}})$; players who state non-equilibrium policies miss their turn.

At the end of the first phase, suppose the players have revealed a set $\Pi$ of policies. Define

$$X_p(\pi) = V_p^\pi(s_{\text{start}}) - V_p^{\text{dis}}(s_{\text{start}})$$
$$\mathbf{U} = \text{convhull} \{\mathbf{X}(\pi) \mid \pi \in \Pi\}$$
$$\underline{\mathbf{U}} = \{\mathbf{u} \geq 0 \mid (\exists \mathbf{v} \in \mathbf{U} \mid \mathbf{u} \leq \mathbf{v})\}$$

where $\mathbf{V}^{\text{dis}}$ is the value function of $\pi^{\text{dis}}$, $X_p(\pi)$ is the excess of policy $\pi$ for player $p$, and $\mathbf{U}$ is the set of feasible excess vectors.

In the second phase, players take turns proposing points $\mathbf{u} \in \underline{\mathbf{U}}$ along with policies or mixtures of policies in $\Pi$ that achieve them. After each proposal, all agents except the pro-

poser decide whether to accept or reject. If everyone accepts, the proposal is implemented: everyone starts executing the agreed equilibrium.

Otherwise, the players who accepted are removed from future negotiation and have their utilities fixed at the proposed levels. Fixing player $p$'s utility at $u_p$ means that all future proposals must give $p$ exactly $u_p$. Invalid proposals cause the proposer to lose her turn. To achieve this, the proposal may require $p$ to voluntarily lower her own utility; this requirement is enforced by the threat that all players will revert to $\pi^{\text{dis}}$ if $p$ fails to act as required.

If at some point we hit the $\epsilon$ chance of having the current round of communication end, all remaining players are assigned their disagreement values. The players execute the last proposed policy $\pi$ (or $\pi^{\text{dis}}$ if there has been no valid proposal), and any player $p$ for whom $V_p^\pi(s_{\text{start}})$ is greater than her assigned utility $u_p$ voluntarily lowers her utility to the correct level. (Again, failure to do so results in all players reverting to $\pi^{\text{dis}}$.)

Under the above protocol, player's preferences are the same as in a Rubinstein game with utility set $\underline{\mathbf{U}}$: because we have assumed that negotiation ends with probability $\epsilon$ after each message, agreeing on $\mathbf{u}$ after $t$ additional steps is exactly as good as agreeing on $\mathbf{u}(1-\epsilon)^{\text{t}}$ now. So with $\epsilon$ sufficiently small, the Rubinstein or Krishna-Serrano results show that rational players will agree on a vector $\mathbf{u} \in \underline{\mathbf{U}}$ which is close to the Nash point $\text{argmax}_{\mathbf{u} \in \underline{\mathbf{U}}} \Pi_{\text{p}} u_{\text{p}}$.

# 4  COMPUTING EQUILIBRIA

In order to use the protocol of Sec. 3 for bargaining in a stochastic game, the players must be able to compute some subgame-perfect equilibria. Computing equilibria is a hard problem, so we cannot expect real agents to find the entire set of equilibria. Fortunately, each player will want to find the equilibria which are most advantageous to herself to influence the negotiation process in her favor. But equilibria which offer other players reasonably high reward have a higher chance of being accepted in negotiation. So, self interest will naturally distribute the computational burden among all the players.

In this section we describe an efficient dynamic-programming algorithm for computing equilibria. The algorithm takes some low-payoff equilibria as input and (usually) outputs higher-payoff equilibria. It is based on the intuition that we can use low-payoff equilibria as enforcement tools: by threatening to switch to an equilibrium that has low value to player $p$, we can deter $p$ from deviating from a cooperative policy.

In more detail, we will assume that we are given $P$ different equilibria $\pi_1^{\text{pun}}, \ldots, \pi_P^{\text{pun}}$; we will use $\pi_p^{\text{pun}}$ to punish player $p$ if she deviates. We can set $\pi_p^{\text{pun}} = \pi^{\text{dis}}$ for all $p$ if $\pi^{\text{dis}}$ is the only equilibrium we know; or, we can use any other equilibrium policies that we happen to have discovered. The algorithm will be most effective when the value of $\pi_p^{\text{pun}}$ to player $p$ is as low as possible in all states.

We will then search for cooperative policies that we can enforce with the given threats $\pi_p^{\text{pun}}$. We will first present an algorithm which pretends that we can efficiently take direct sums and convex hulls of arbitrary sets. This algorithm is impractical, but finds all enforceable value vectors. We will then turn it into an approximate algorithm which uses finite data structures to represent the set-valued variables. As we allow more and more storage for each set, the approximate algorithm will approach the exact one; and in any case the result will be a set of equilibria which the agents can execute.

## 4.1  THE EXACT ALGORITHM

Our algorithm maintains a set of value vectors $\mathbf{V}(s)$ for each state $s$. It initializes $\mathbf{V}(s)$ to a set which we know contains the value vectors for all equilibrium policies. It then refines $\mathbf{V}$ by dynamic programming: it repeatedly attempts to improve the set of values at each state by backing up all of the joint actions, excluding joint actions from which some agent has an incentive to deviate.

In more detail, we will compute $V_p^{\text{dis}}(s) \equiv V_p^{\pi_{\text{dis}}}(s)$ for all $s$ and $p$ and use the vector $\mathbf{V}^{\text{dis}}(s)$ in our initialization. (Recall that we have defined $V_p^\pi(s)$ for a nonstationary policy $\pi$ as the value of $\pi$ if $s$ were the start state.) We also need the values of the punishment policies for

*Initialization*
**for** $s \in S$
    $\mathbf{V}(s) \leftarrow \{\mathbf{V} \mid V_p^{\mathrm{dis}}(s) \leq V_p \leq R_{\max}/(1-\gamma)\}$
**end**

*Repeat until converged*
**for** iteration $\leftarrow 1, 2, \ldots$
    **for** $s \in S$
      *Compute value vector set for each joint action,*
        *then throw away unenforceable vectors*
      **for** $a \in A$
        $\mathbf{Q}(s,a) \leftarrow \{\mathbf{R}(s,a)\} + \gamma \sum_{s' \in S} T(s,a)(s')\mathbf{V}(s')$
        $\mathbf{Q}(s,a) \leftarrow \{\mathbf{Q} \in \mathbf{Q}(s,a) \mid \mathbf{Q} \geq \mathbf{V}^{\mathrm{dev}}(s,a)\}$
      **end**
      *We can now randomize among joint actions*
      $\mathbf{V}(s) \leftarrow \mathrm{convhull} \bigcup_a \mathbf{Q}(s,a)$
    **end**
**end**

Figure 2: Dynamic programming using exact operations on sets of value vectors

their corresponding players, $V_p^{\mathrm{pun}}(s) \equiv V_p^{\pi_p^{\mathrm{pun}}}(s)$ for all $p$ and $s$. Given these values, define

$$Q_p^{\mathrm{dev}}(s,a) = R_p(s,a) + \gamma \sum_{s' \in S} T(s,a)(s')V_p^{\mathrm{pun}}(s') \tag{1}$$

to be the value to player $p$ of playing joint action $a$ from state $s$ and then following $\pi_p^{\mathrm{pun}}$ forever after.

From the above $Q_p^{\mathrm{dev}}$ values we can compute player $p$'s value for deviating from an equilibrium which recommends action $a$ in state $s$: it is $Q_p^{\mathrm{dev}}(s,a')$ for the best possible deviation $a'$, since $p$ will get the one-step payoff for $a'$ but be punished by the rest of the players starting on the following time step. That is,

$$V_p^{\mathrm{dev}}(s,a) = \max_{a_p' \in A_p} Q_p^{\mathrm{dev}}(s, a_1 \times \ldots \times a_p' \times \ldots \times a_P) \tag{2}$$

$V_p^{\mathrm{dev}}(s,a)$ is the value we must achieve for player $p$ in state $s$ if we are planning to recommend action $a$ and punish deviations with $\pi_p^{\mathrm{pun}}$: if we do not achieve this value, player $p$ would rather deviate and be punished.

Our algorithm is shown in Fig. 2. After $k$ iterations, each vector in $\mathbf{V}(s)$ corresponds to a $k$-step policy in which no agent ever has an incentive to deviate. In the $k+1$st iteration, the first assignment to $\mathbf{Q}(s,a)$ computes the value of performing action $a$ followed by any $k$-step policy. The second assignment throws out the pairs $(a, \pi)$ for which some agent would want to deviate from $a$ given that the agents plan to follow $\pi$ in the future. And the convex hull accounts for the fact that, on reaching state $s$, we can select an action $a$ and future policy $\pi$ at random from the feasible pairs.[2] Proofs of convergence and correctness of the exact algorithm are in the technical report [13].

Of course, we cannot actually implement the algorithm of Fig. 2, since it requires variables whose values are convex sets of vectors. But, we can approximate $\mathbf{V}(s)$ by choosing a finite set of witness vectors $\mathbf{W} \subset \mathbb{R}^P$ and storing $\mathbf{V}(s, \mathbf{w}) = \arg\max_{\mathbf{v} \in \mathbf{V}(s)}(\mathbf{v} \cdot \mathbf{w})$ for each $\mathbf{w} \in \mathbf{W}$. $\mathbf{V}(s)$ is then approximated by the convex hull of $\{\mathbf{V}(s, \mathbf{w}) \mid \mathbf{w} \in \mathbf{W}\}$. If $\mathbf{W}$ samples the $P$-dimensional unit hypersphere densely enough, the maximum possible approximation error will be small. (In practice, each agent will probably want to pick $\mathbf{W}$ differently, to focus her computation on policies in the portion of the Pareto frontier where her own utility is relatively high.) As $|\mathbf{W}|$ increases, the error introduced at each step will go to zero. The approximate algorithm is given in more detail in the technical report [13].

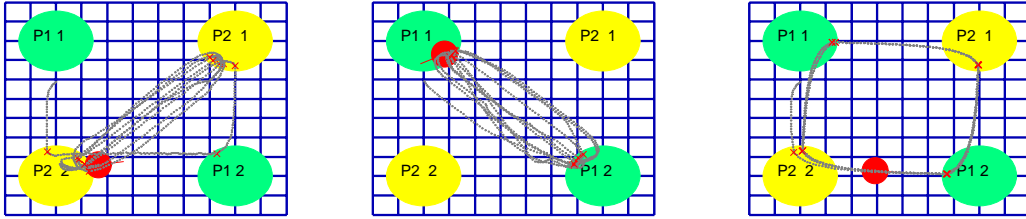

Figure 3: Execution traces for our motion planning example. Left and Center: with 2 witness vectors , the agents randomize between two selfish paths. Right: with 4–32 witnesses, the agents find a cooperative path. Steps where either player gets a goal are marked with ×.

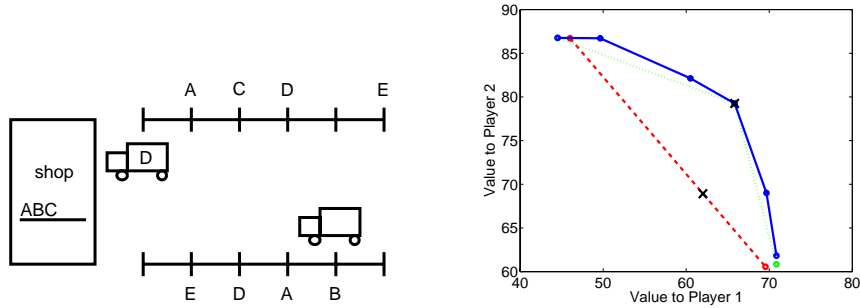

Figure 4: Supply chain management problem. In the left figure, Player 1 is about to deliver part $D$ to the shop, while player 2 is at the warehouse which sells $B$. The right figure shows the tradeoff between accuracy and computation time. The solid curve is the Pareto frontier for $s_{\text{start}}$, as computed using 8 witnesses per state. The dashed and dotted lines were computed using 2 and 4 witnesses, respectively. Dots indicate computed value vectors; × marks indicate the Nash points.

## 5 EXPERIMENTS

We tested our value iteration algorithm and negotiation procedure on two robotic planning domains: a joint motion planning problem and a supply-chain management problem.

In our motion planning problem (Fig. 3), two players together control a two-wheeled robot, with each player picking the rotational velocity for one wheel. Each player has a list of goal landmarks which she wants to cycle through, but the two players can have different lists of goals. We discretized states based on $X, Y, \theta$ and the current goals, and discretized actions into stop, slow $(0.45\frac{m}{s})$, and fast $(0.9\frac{m}{s})$, for 9 joint actions and about 25,000 states. We discretized time at $\Delta t = 1s$, and set $\gamma = 0.99$. For both the disagreement policy and all punishment policies, we used "always stop," since by keeping her wheel stopped either player can prevent the robot from moving. Planning took a few hours of wall clock time on a desktop workstation for 32 witnesses per state.

Based on the planner's output, we ran our negotiation protocol to select an equilibrium. Fig. 3 shows the results: with limited computation the players pick two selfish paths and randomize equally between them, while with more computation they find the cooperative path. Our experiments also showed that limiting the computation available to one player allows the unrestricted player to reveal only some of the equilibria she knows about, tilting the outcome of the negotiation in her favor (see the technical report [13] for details).

For our second experiment we examined a more realistic supply-chain problem. Here each player is a parts supplier competing for the business of an engine manufacturer. The manufacturer doesn't store items and will only pay for parts which can be used immediately. Each player controls a truck which moves parts from warehouses to the assembly shop; she pays for parts when she picks them up, and receives payment on delivery. Each player gets

parts from different locations at different prices and no one player can provide all of the parts the manufacturer needs.

Each player's truck can be at six locations along a line: four warehouse locations (each of which provides a different type of part), one empty location, and the assembly shop. Building an engine requires five parts, delivered in the order $A, \{B, C\}, D, E$ (parts $B$ and $C$ can arrive in either order). After $E$, the manufacturer needs $A$ again. Players can move left or right along the line at a small cost, or wait for free. They can also buy parts at a warehouse (dropping any previous cargo), or sell their cargo if they are at the shop and the manufacturer wants it. Each player can only carry one part at a time and only one player can make a delivery at a time. Finally, any player can retire and sell her truck; in this case the game ends and all players get the value of their truck plus any cargo. The disagreement policy is for all players to retire at all states. Fig. 4 shows the computed sets $\mathbf{V}(s_{\text{start}})$ for various numbers of witnesses. The more witnesses we use, the more accurately we represent the frontier, and the closer our final policy is to the true Nash point.

All of the policies computed are "intelligent" and "cooperative": a human observer would not see obvious ways to improve them, and in fact would say that they look similar despite their differing payoffs. Players coordinate their motions, so that one player will drive out to buy part $E$ while the other delivers part $D$. They sit idle only in order to delay the purchase of a part which would otherwise be delivered too soon.

## 6   CONCLUSION

Real-world planning problems involve negotiation among multiple agents with varying goals. To take all agents incentives into account, the agents should find and agree on Pareto-dominant subgame-perfect Nash equilibria. For this purpose, we presented efficient planning and negotiation algorithms for general-sum stochastic games, and tested them on two robotic planning problems.

## Footnotes

[1] We gratefully acknowledge help and comments from Ron Parr on this research. This work was supported in part by DARPA contracts HR0011-06-0023 (the CS2P program) and 55-00069 (the RADAR program). All opinions, conclusions, and errors are our own.

[2]It is important for this randomization to occur *after* reaching state $s$ to avoid introducing incentives to deviate, and it is also important for the randomization to be public.

## References

[1] V. Conitzer and T. Sandholm. Complexity results about Nash equilibria. Technical Report CMU-CS-02-135, School of Computer Science, Carnegie-Mellon University, 2002.

[2] D. P. Bertsekas. *Dynamic Programming and Optimal Control*. Athena Scientific, Massachusetts, 1995.

[3] Prajit K. Dutta. A folk theorem for stochastic games. *Journal of Economic Theory*, 66:1–32, 1995.

[4] Yevgeniy Dodis, Shai Halevi, and Tal Rabin. A cryptographic solution to a game theoretic problem. In *Lecture Notes in Computer Science*, volume 1880, page 112. Springer, Berlin, 2000.

[5] Michael L. Littman and Peter Stone. A polynomial-time Nash equilibrium algorithm for repeated games. In *ACM Conference on Electronic Commerce*, pages 48–54. ACM, 2003.

[6] E. Hansen, D. Bernstein, and S. Zilberstein. Dynamic programming for partially observable stochastic games. In *Proceedings of the Nineteenth National Conference on Artificial Intelligence*, pages 709–715, 2004.

[7] Ulrich Doraszelski and Kenneth L. Judd. Avoiding the curse of dimensionality in dynamic stochastic games. *NBER Technical Working Paper No. 304*, January 2005.

[8] R. Brafman and M. Tennenholtz. Efficient learning equilibrium. *Artificial Intelligence*, 2004.

[9] D Fudenberg and E. Maskin. The folk theorem in repeated games with discounting or with incomplete information. *Econometrica*, 1986.

[10] David Levine. The castle on the hill. *Review of Economic Dynamics*, 3(2):330–337, 2000.

[11] Ariel Rubinstein. Perfect equilibrium in a bargaining model. *Econometrica*, 50(1):97–109, 1982.

[12] V. Krishna and R. Serrano. Multilateral bargaining. *Review of Economic Studies*, 1996.

[13] Chris Murray and Geoffrey J. Gordon. Multi-robot negotiation: approximating the set of subgame perfect equilibria in general-sum stochastic games. Technical Report CMU-ML-06-114, Carnegie Mellon University, 2006.
